# Statistical Reliability of a Blowfly Movement-Sensitive Neuron

**Rob de Ruyter van Steveninck** *
Biophysics Group,
Rijksuniversiteit Groningen,
Groningen, The Netherlands

**William Bialek**
NEC Research Institute
4 Independence Way,
Princeton, NJ 08540

## Abstract

We develop a model-independent method for characterizing the reliability of neural responses to brief stimuli. This approach allows us to measure the discriminability of similar stimuli, based on the real-time response of a single neuron. Neurophysiological data were obtained from a movement-sensitive neuron (H1) in the visual system of the blowfly *Calliphora erythrocephala*. Furthermore, recordings were made from blowfly photoreceptor cells to quantify the signal to noise ratios in the peripheral visual system. As photoreceptors form the input to the visual system, the reliability of their signals ultimately determines the reliability of any visual discrimination task. For the case of movement detection, this limit can be computed, and compared to the H1 neuron's reliability. Under favorable conditions, the performance of the H1 neuron closely approaches the theoretical limit, which means that under these conditions the nervous system adds little noise in the process of computing movement from the correlations of signals in the photoreceptor array.

## 1 INTRODUCTION

In the 1940s and 50s, several investigators realized that understanding the reliability of computation in the nervous system posed significant theoretical challenges. Attempts to perform reliable computations with the available electronic computers

certainly posed serious practical problems, and the possibility that the problems of natural and artificial computing are related was explored. Guided by the practical problems of electronic computing, von Neumann (1956) formulated the theoretical problem of "reliable computation with unreliable components". Many authors seem to take as self-evident the claim that this is a problem faced by the nervous system as well, and indeed the possibility that the brain may implement novel solutions to this problem has been at least a partial stimulus for much recent research. The qualitative picture adopted in this approach is of the nervous system as a highly interconnected network of rather noisy cells, in which meaningful signals are represented only by large numbers of neural firing events averaged over numerous redundant neurons. Neurophysiological experiments seem to support this view: If the same stimulus is presented repeatedly to a sensory system, the responses of an individual afferent neuron differ for each presentation. This apparently has led to a widespread belief that neurons are inherently noisy, and ideas of redundancy and averaging pervade much of the literature. Significant objections to this view have been raised, however (*cf.* Bullock 1970).

As emphasized by Bullock (*loc.cit*), the issue of reliability of the nervous system is a quantitative one. Thus, the first problem that should be overcome is to find a way for its measurement. This paper focuses on a restricted, but basic question, namely the reliability of a single neuron, much in the spirit of previous work (cf. Barlow and Levick 1969, Levick et al. 1983, Tolhurst at al. 1983, Parker and Hawken 1985). Here the methods of analysis used by these authors are extended in an attempt to describe the neuron's reliability in a way that is as model-independent as possible.

The second–conceptually more difficult–problem, is summarized cogently in Bullock's words, "how reliable is reliable?". Just quantifying reliability is not enough, and the qualitative question of whether redundancy, averaging, multiplexing, or yet more exotic solutions to von Neumann's problem are relevant to the operation of the nervous system hinges on a quantitative comparison of reliability at the level of single cells with the reliability for the whole system. Broadly speaking, there are two ways to make such a comparison: one can compare the performance of the single cell either with the output or with the input of the whole system. As to the first possibility, if a single cell responds to a certain stimulus as reliably as the animal does in a behavioral experiment, it is difficult to imagine why multiple redundant neurons should be used to encode the same stimulus. Alternatively, if the reliability of a single neuron were to approach the limits set by the sensory periphery, there seems to be little purpose for the nervous system to use functional duplicates of such a cell, and the key theoretical problem would be to understand how such optimal processing is implemented. Here we will use the latter approach.

We first quantify the reliability of response of H1, a wide-field movement-sensitive neuron in the blowfly visual system. The method consists essentially of a direct application of signal detection theory to trains of neural impulses generated by brief stimuli, using methods familiar from psychophysics to quantify discriminability. Next we characterize signal transfer and noise in the sensory periphery–the photoreceptor cells of the compound eye–and we compare the reliability of information coded in H1 with the total amount of sensory information available at the input.

## 2   PREPARATION, STIMULATION AND RECORDING

Experiments were performed on female wild-type blowfly *Calliphora erythrocephala*. Spikes from H1 were recorded extracellularly with a tungsten microelectrode, their arrival times being digitized with 50 $\mu$s resolution. The fly watched a binary random-bar pattern (bar width 0.029° visual angle, total size $(30.5°)^2$) displayed on a CRT. Movement steps of 16 different sizes (integer multiples of 0.12°) were generated by custom-built electronics, and presented at 200 ms intervals in the neuron's preferred direction. The effective duration of the experiment was 11 hours, during which time about $10^6$ spikes were recorded over 12552 presentations of the 16-step stimulus sequence.

Photoreceptor cells were recorded intracellularly while stimulated by a spatially homogeneous field, generated on the same CRT that was used for the H1 experiments. The CRT's intensity was modulated by a binary pseudo-random waveform, time sampled at 1 ms. The responses to 100 stimulus periods were averaged, and the cell's transfer function was obtained by computing the ratio of the Fourier transform of the averaged response to that of the stimulus signal. The cell's noise power spectrum was obtained by averaging the power spectra of the 100 traces of the individual responses with the average response subtracted.

## 3   DATA ANALYSIS

### 3.1   REPRESENTATION OF STIMULUS AND RESPONSE

A single movement stimulus consisted of a sudden small displacement of a wide-field pattern. Steps of varying sizes were presented at regular time-intervals, long enough to ensure that responses to successive stimuli were independent. In the analysis we consider the stimulus to be a point event in time, parametrized by its step size $\alpha$.

The neuron's signal is treated as a stochastic point process, the parameters of which depend on the stimulus. Its statistical behavior is described by the conditional probability $P(r|\alpha)$ of finding a response $r$, given that a step of size $\alpha$ was presented. From the experimental data we estimate $P(r|\alpha)$ for each step size separately. To represent a single response $r$, time is divided in discrete bins of width $\Delta t = 2$ ms. Then $r$ is described by a firing pattern, which is just a vector $\vec{q} = [q_0, q_1, ..]$ of binary digits $q_k(k = 0, n - 1)$, where $q_k = 1$ and $q_k = 0$ respectively signify the presence or the absence of a spike in time bin $k$ (cf. Eckhorn and Pöpel 1974). No response is found within a latency time $t_{lat}=15$ ms after stimulus presentation; spikes fired within this interval are due to spontaneous activity and are excluded from analysis, so $k = 0$ corresponds to 15 ms after stimulus presentation.

The probability distribution of firing patterns, $P(\vec{q}|\alpha)$, is estimated by counting the number of occurrences of each realization of $\vec{q}$ for a large number of presentations of $\alpha$. This distribution is described by a tree which results from ordering all recorded firing patterns according to their binary representation, earlier times corresponding to more-significant bits. Graphical representations of two such trees are shown in Fig. 1. In constructing a tree we thus perform two operations on the raw spike data: first, individual response patterns are represented in discrete time bins $\Delta t$, and second, a permutation is performed on the set of discretized patterns to order

them according to their binary representation. No additional assumptions are made about the way the signal is encoded by the neuron. This approach should therefore be quite powerful in revealing any subtle "hidden code" that the neuron might use. As the number of branches in the tree grows exponentially with the number of time bins $n$, many presentations are needed to describe the tree over a reasonable time interval, and here we use $n = 13$.

## 3.2   COMPUTATION OF DISCRIMINABILITY

To quantify the performance of the neuron, we compute the discriminability of two nearly equal stimuli $\alpha_1$ and $\alpha_2$, based on the difference in neural response statistics described by $P(r|\alpha_1)$ and $P(r|\alpha_2)$. The probability of correct decisions is maximized if one uses a maximum likelihood decision rule, so that in the case of equal prior probabilities the outcome is $\alpha_1$ if $P(r_{obs}|\alpha_1) > P(r_{obs}|\alpha_2)$, and vice versa. On average, the probability of correctly identifying step $\alpha_1$ is then:

$$P_c(\alpha_1) = \sum_{\{r\}} P(r|\alpha_1) \cdot H[P(r|\alpha_1) - P(r|\alpha_2)], \qquad (1)$$

where $H(.)$ is the Heaviside step function and the summation is over the set of all possible responses $\{r\}$. An interchange of indices 1 and 2 in this expression yields the formula for correct identification of $\alpha_2$. The probability of making correct judgements over an entire experiment in which $\alpha_1$ and $\alpha_2$ are equiprobable is then simply $P_c(\alpha_1, \alpha_2) = [P_c(\alpha_1) + P_c(\alpha_2)]/2$, which from now on will be referred to as $P_c$.

This analysis is essentially that for a "two-alternative forced-choice" psychophysical experiment. For convenience we convert $P_c$ into the discriminability parameter $d'$, familiar from psychophysics (Green and Swets 1966), which is the signal-to-noise ratio (difference in mean divided by the standard deviation) in the equivalent equal-variance Gaussian decision problem.

Using the firing-pattern representation, $r = \vec{q}$, and computing $d'$ for successive subvectors of $\vec{q}$ with elements $m = 0, .., k$ and $k = 0, .., n - 1$, we compute $P_c$ for different values of $k$ and from that obtain $d'(k)$, the discriminability as a function of time.

## 3.3   THEORETICAL LIMITS TO DISCRIMINATION

For the simple stimuli used here it is relatively easy to determine the theoretical limit to discrimination based on the photoreceptor signal quality. For the computation of this limit we use Reichardt's (1957) correlation model of movement detection. This model has been very successful in describing a wide variety of phenomena in biological movement detection, both in fly (Reichardt and Poggio 1976), and in humans (van Santen and Sperling 1984). Also, correlation-like operations can be proved to be optimal for the extraction of movement information at low signal to noise ratio (Bialek 1990). The measured signal transfer of the photoreceptors, combined with the known geometry of the stimulus and the optics of the visual system determine the signal input to the model. The noise input is taken directly

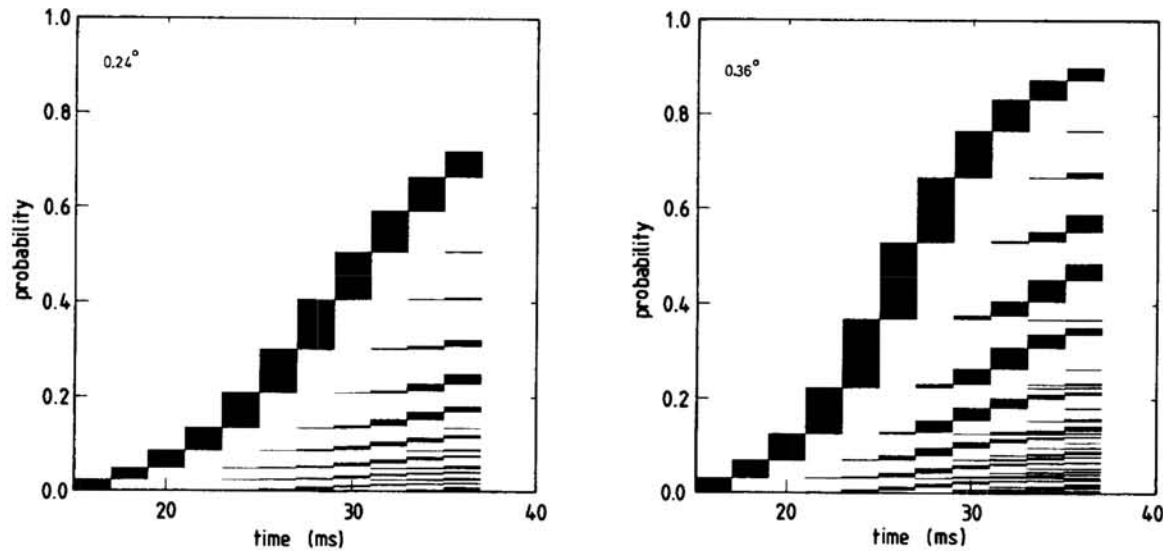

Figure 1: Representation of the firing pattern distributions for steps of 0.24° and 0.36°. Here only 11 time bins are shown.

from the measured photoreceptor noise power spectrum. Details of this computation are given in de Ruyter van Steveninck (1986).

## 3.4 ERROR ANALYSIS AND DATA REQUIREMENTS

The effects of the approximation due to time-discretization can be assessed by varying the binwidth. It turns out that the results do not change appreciably if the bins are made smaller than 2 ms. Furthermore, if the analysis is to make sense, stationarity is required, i.e. the probability distribution from which responses to a certain stimulus are drawn should be invariant over the course of the experiment. Finally, the distributions, being computed from a finite sample of responses, are subject to statistical error. The statistical error in the final result was estimated by partitioning the data and working out the values of $P_c$ for these partitions separately. The statistical variations in $P_c$ were of the order of 0.01 in the most interesting region of values of $P_c$, i.e. from 0.6 to 0.9. This results in a typical statistical error of 0.05 in $d'$. In addition, this analysis revealed no significant trends with time, so we may assume stationarity of the preparation.

## 4  RESULTS

### 4.1  STEP SIZE DISCRIMINATION BY THE H1 NEURON

Although 16 different step sizes were used, we limit the presentation here to steps of 0.24° and 0.36°; binary trees representing the two firing-pattern distributions are shown in Fig. 1. The first time bin describes the probabilities of two possible events: either a spike was fired (black) or not (white), and these two probabilities add up to unity. The second time bin describes the four possible combinations of finding

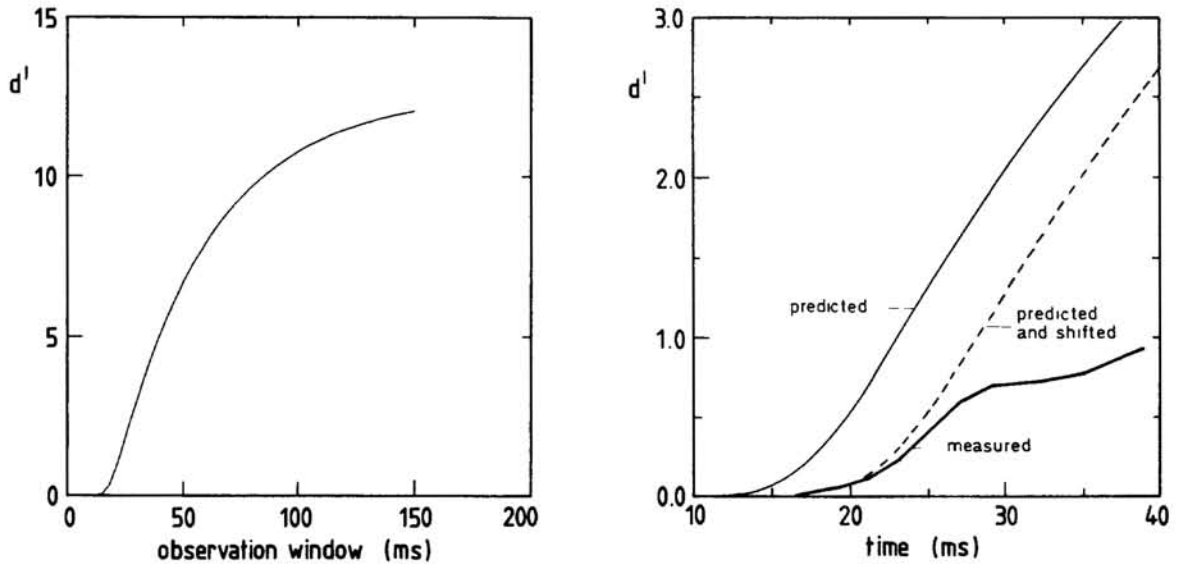

Figure 2: Left: Discrimination performance of an ideal movement detector. See text for further details. Right: comparison of the theoretical and the measured values of $d'(t)$. Fat line: measured performance of H1. Thin solid line: predicted performance, taken from the left figure. Dashed line: the same curve shifted by 5 ms to account for latency time in the pathway from photoreceptor to H1. This time interval was determined independently with powerful movement stimuli.

or not finding a spike in bin 2 combined with finding or not finding a spike in bin 1, and so on. The figure shows that the probability of firing a spike in time bin 1 is slightly higher for the larger step. From above we compute $P_c$, the probability of correct identification, in a task where the choice is between step sizes of 0.24° and 0.36° with equal prior probabilities. The decision rule is simple: if a spike is fired in bin 1, choose the larger, otherwise choose the smaller step. In the same fashion we apply this procedure to the following time bin, with four response categories and so on. The value of $d'$ computed from $P_c$ for this step size pair as a function of time is given by the fat line at the right in Fig. 2.

## 4.2    LIMITS SET BY PHOTORECEPTOR SIGNALS

Figure 2 (left) shows the limit to movement detection computed for an array of 2650 Reichardt correlators stimulated with a step size difference of 0.12°, conforming to the experimental conditions. Comparing the performance of H1 to this result (the fat and the dashed lines in Fig. 2, right), we see that the neuron follows the limit set by the sensory periphery from about 18 to 28 ms after stimulus presentation. So, for this time window the randomness of H1's response is determined primarily by photoreceptor noise. Up to about 20 Hz, the photoreceptor signal-to-noise ratio closely approached the limit set by the random arrival of photons at the photoreceptors at a rate of about $10^4$ effective conversions/s. Hence most of the randomness in the spike train was caused by photon shot noise.

# 5   DISCUSSION

The approach presented here gives us estimates for the reliability of a single neuron in a well-defined, though restricted experimental context. In addition the theoretical limits to the reliability of movement-detection are computed. Comparing these two results we find that H1 in these conditions uses essentially all of the movement information available over a 10 ms time interval. Further analysis shows that this information is essentially contained in the time of firing of the first spike. The plateau in the measured $d'(t)$ between 28 and 34 ms presumably results from effects of refractoriness, and the subsequent slight rise is due to firing of a second spike.

Thus, a step size difference of 0.12° can be discriminated with $d'$ close to unity, using the timing information of just one spike from one neuron. For the blowfly visual system this angular difference is of the order of one-tenth of the photoreceptor spacing, well within the hyperacuity regime (cf. Parker and Hawken 1985).

It should not be too surprising that the neuron performs well only over a short time interval and does not reach the values for $d'$ computed from the model at large delays (Fig. 2, left): The experimental stimulus is not very natural, and in real-life conditions the fly is likely to see movement changing continuously. (Methods for analyzing responses to continuous movement are treated in de Ruyter van Steveninck and Bialek 1988, and in Bialek et al. 1991.) In such circumstances it might be better not to wait very long to get an accurate estimate of the stimulus at one point in time, but rather to update rough estimates as fast as possible. This would favor a coding principle where successive spikes code independent events, which may explain that the plateau in the measured $d'(t)$ starts at about the point where the computed $d'(t)$ has maximal slope. Such a view is supported by behavioral evidence: A chasing fly tracks the leading fly with a delay of about 30 ms (Land and Collett 1974), corresponding to the time at which the measured $d'(t)$ levels off.

In conclusion we can say that in the experiment, for a limited time window the neuron effectively uses all information available at the sensory periphery. Peripheral noise is in turn determined by photon shot noise so that the reliability of H1's output is set by the physics of its inputs. There is no neuro-anatomical or neurophysiological evidence for massive redundancy in arthropod nervous systems. More specifically, for the fly visual system, it is known that H1 is unique in its combination of visual field and preferred direction of movement (Hausen 1982), and from the results presented here we may begin to understand why: It just makes little sense to use functional duplicates of any neuron that performs almost perfectly when compared to the noise levels inherently present in the stimulus. It remains to be seen to what extent this conclusion can be generalized, but one should at least be cautious in interpreting the variability of response of a single neuron in terms of noise generated by the nervous system itself.

## Footnotes

*present address: University Hospital Groningen, Dept. of Audiology, POB 30.001, NL 9700RB Groningen, The Netherlands

## References

Barlow HB, Levick WR (1969) Three factors limiting the reliable detection of light by retinal ganglion cells of the cat. J Physiol 200:1-24.

Bialek W (1990) Theoretical physics meets experimental neurobiology. In Jen E (ed.) *1989 Lectures in Complex Systems, SFI Studies in the Sciences of Complexity,*

*Lect. Vol. II*, pp. 513-595. Addison-Wesley, Menlo Park CA.

Bialek W, Rieke F, de Ruyter van Steveninck RR, Warland D (1991) Reading a neural code. Science **252**:1854-1857.

Bullock TH (1970) The reliability of neurons. J Gen Physiol **55**:565-584.

Eckhorn R, Pöpel B (1974) Rigorous and extended application of information theory to the afferent visual system of the cat. I Basic concepts. Kybernetik **16**:191-200.

Green DM, Swets JA (1966) *Signal detection theory and psychophysics.* Wiley, New York.

Hausen K (1982) Motion sensitive interneurons in the optomotor system of the fly. I. The horizontal cells: Structure and signals. Biol Cybern **45**:143-156.

Land MF, Collett TS (1974) Chasing behaviour of houseflies (*Fannia canicularis*). A description and analysis. J Comp Physiol **89**:331-357.

Levick WR, Thibos LN, Cohn TE, Catanzaro D, Barlow HB (1983) Performance of cat retinal ganglion cells at low light levels. J Gen Physiol **82**:405-426.

Neumann J von (1956) Probabilistic logics and the synthesis of reliable organisms from unreliable components. In Shannon CE and McCarthy J (eds.) *Automata Studies*, Princeton University Press, Princeton NJ, 43-98.

Parker A, Hawken M (1985) Capabilities of monkey cortical cells in spatial-resolution tasks. J Opt Soc Am **A2**:1101-1114.

Reichardt W (1957) Autokorrelations-Auswertung als Funktionsprinzip des Zentralnervensystems. Z Naturf **12b**:448-457.

Reichardt W, Poggio T (1976) Visual control of orientation behaviour in the fly, Part I. A quantitative analysis. Q Rev Biophys **9**:311-375.

de Ruyter van Steveninck RR (1986) *Real-time performance of a movement-sensitive neuron in the blowfly visual system.* Thesis, Rijksuniversiteit Groningen, the Netherlands.

de Ruyter van Steveninck RR, Bialek W (1988) Real-time performance of a movement-sensitive neuron in the blowfly visual system: coding and information transfer in short spike sequences. Proc R Soc Lond B **234**: 379-414.

van Santen JPH, Sperling G (1984) Temporal covariance model of human motion perception. J Opt Soc Am **A1**:451-473.

Tolhurst DJ, Movshon JA, Dean AF (1983) The statistical reliability of signals in single neurons in cat and monkey visual cortex. Vision Res **23**: 775-785.